# Optimal Depth Neural Networks for Multiplication and Related Problems

**Kai-Yeung Siu**
Dept. of Electrical & Comp. Engineering
University of California, Irvine
Irvine, CA 92717

**Vwani Roychowdhury**
School of Electrical Engineering
Purdue University
West Lafayette, IN 47907

## Abstract

An artificial neural network (ANN) is commonly modeled by a threshold circuit, a network of interconnected processing units called linear threshold gates. The depth of a network represents the number of unit delays or the time for parallel computation. The size of a circuit is the number of gates and measures the amount of hardware. It was known that traditional logic circuits consisting of only unbounded fan-in AND, OR, NOT gates would require at least $\Omega(\log n / \log\log n)$ depth to compute common arithmetic functions such as the product or the quotient of two $n$-bit numbers, unless we allow the size (and fan-in) to increase exponentially (in $n$). We show in this paper that ANNs can be much more powerful than traditional logic circuits. In particular, we prove that that iterated addition can be computed by depth-2 ANN, and multiplication and division can be computed by depth-3 ANNs with polynomial size and polynomially bounded integer weights, respectively. Moreover, it follows from known lower bound results that these ANNs are optimal in depth. We also indicate that these techniques can be applied to construct polynomial-size depth-3 ANN for powering, and depth-4 ANN for multiple product.

## 1  Introduction

Recent interest in the application of artificial neural networks [10, 11] has spurred research interest in the theoretical study of such networks. In most models of neural networks, the basic processing unit is a Boolean gate that computes a linear

threshold function, or an analog element that computes a sigmoidal function. Artificial neural networks can be viewed as circuits of these processing units which are massively interconnected together.

While neural networks have found wide application in many areas, the behavior and the limitation of these networks are far from being understood. One common model of a neural network is a *threshold circuit*. Incidentally, the study of threshold circuits, motivated by some other complexity theoretic issues, has also gained much interest in the area of computer science. Threshold circuits are Boolean circuits in which each gate computes a linear threshold function, whereas in the classical model of *unbounded fan-in* Boolean circuits only AND, OR, NOT gates are allowed. A Boolean circuit is usually arranged in layers such that all gates in the same layer are computed concurrently and the circuit is computed layer by layer in some increasing *depth* order. We define the *depth* as the number of layers in the circuit. Thus each layer represents a unit delay and the depth represents the overall delay in the computation of the circuit.

## 2    Related Work

Theoretical computer scientists have used *unbounded fan-in* Boolean circuits as a model to understand fundamental issues of parallel computation. To be more specific, this computational model should be referred to as *unbounded fan-in* parallelism, since the number of inputs to each gate in the Boolean circuit is not bounded by a constant. The theoretical study of unbounded fan-in parallelism may give us insights into devising faster algorithms for various computational problems than would be possible with bounded fan-in parallelism. In fact, any nondegenerate Boolean function of $n$ variables requires at least $\Omega(\log n)$ depth to compute in a bounded fan-in circuit. On the other hand, in some practical situations, (for example large fan-in circuits such as programmable logic arrays (PLAs) or multiple processors simultaneously accessing a shared bus), unbounded fan-in parallelism seems to be a natural model. For example, a PLA can be considered as a depth-2 AND/OR circuit.

In the Boolean circuit model, the amount of resources is usually measured by the number of gates, and is considered to be 'reasonable' as long as it is bounded by a polynomial (as opposed to exponential) in the number of the inputs. For example, a Boolean circuit for computing the sum of two $n$-bit numbers with $O(n^3)$ gates is 'reasonable', though circuit designers might consider the size of the circuit impractical for moderately large $n$. One of the most important theoretical issues in parallel computation is the following: *Given that the number of gates in the Boolean circuit is bounded by a polynomial in the size of inputs, what is the minimum depth (i.e. number of layers) that is needed to compute certain functions?*

A first step toward answering this important question was taken by Furst et al. [4] and independently by Ajtai [2]. It follows from their results that for many basic functions, such as the *parity* and the *majority* of $n$ Boolean variables, or the multiplication of two $n$-bit numbers, any constant depth (*i.e.* independent of $n$) classical Boolean circuit of unbounded fan-in AND/OR gates computing these functions must have more than a polynomial (in $n$) number of gates. This lower bound on the size was subsequently improved by Yao [18] and Hastad [7]; it was proved that

indeed an exponential number of AND/OR gates are needed. So functions such as *parity* and *majority* are computationally 'hard' with respect to constant depth and polynomial size classical Boolean circuits. Another way of interpreting these results is that circuits of AND/OR gates computing these 'hard' functions which use polynomial amount of chip area must have *unbounded delay* (*i.e.* delay that increases with $n$). In fact, the lower bound results imply that the minimum possible delay for multipliers (with polynomial number of AND/OR gates) is $\Omega(\log n/\log\log n)$. These results also give theoretical justification why it is impossible for circuit designers to implement fast parity circuit or multiplier in small chip area using AND, OR gates as the basic building blocks.

One of the 'hard' functions mentioned above is the *majority* function, a special case of a threshold function in which the *weights* or parameters are restricted. A natural extension is to study Boolean circuits that contain *majority gates*. This type of Boolean circuit is called a threshold circuit and is believed to capture some aspects of the computation in our brain [12]. In the rest of the paper, the term 'neural networks' refers to the threshold circuits model.

With the addition of majority gates, the resulting Boolean circuit model seems much more powerful than the classical one. Indeed, it was first shown by Muroga [13] three decades ago that any symmetric Boolean function (*e.g.* parity) can be computed by a two-layer neural network with $(n + 1)$ gates. Recently, Chandra et al. [3] showed that multiplication of two $n$-bit numbers and sorting of $n$ $n$-bit numbers can be computed by neural networks with 'constant' depth and polynomial size. These 'constants' have been significantly reduced by Siu and Bruck [14, 15] to 4 in both cases, whereas a lower bound of depth-3 was proved by Hajnal et al. [6] in the case of multiplication. It is now known [8] that the size of the depth-4 neural networks for multiplication can be reduced to $O(n^2)$. However, the existence of depth-3 and polynomial-size neural networks for multiplication was left as an open problem [6, 5, 15] since the lower bound result in [6]. In [16], some depth-efficient neural networks were constructed for division and related arithmetic problems; the networks in [16] do not have optimal depth.

Our main contribution in this paper is to show that small constant depth neural networks for multiplication, division and related problems can be constructed. For the problems such as iterated addition, multiplication, and division, the neural networks constructed can be shown to have optimal depth. These results have the following implication on their practical significance: *Suppose we can use analog devices to build threshold gates with a cost (in terms of delay and chip area) that is comparable to that of AND, OR, logic gates, then we can compute many basic functions much faster than using traditional circuits.* Clearly, the particular weighting of depth, fan-in, and size that gives a realistic measure of a network's cost and speed depends on the technology used to build it. One case where circuit depth would seem to be the most important parameter is when the circuit is implemented using optical devices. We refer those who are interested in the optical implementation of neural networks to [1].

Due to space limitations, we shall only state some of the important results; further results and detailed proofs will appear in the journal version of this paper [17].

## 3    Main Results

**Definition 1**    Given $n$ $n$-bit integers, $z_i = \sum_{j=0}^{n-1} z_{i,j} 2^j$, $i = 1, ..., n$, $z_{i,j} \in \{0, 1\}$, We define *iterated addition* to be the problem of computing the $(n + \log n)$-bit sum $\sum_{i=1}^{n} z_i$ of the $n$ integers.

**Definition 2**    Given 2 $n$-bit integers, $x = \sum_{j=0}^{n-1} x_j 2^j$ and $y = \sum_{j=0}^{n-1} y_j 2^j$. We define *multiplication* to be the problem of computing the $(2n)$-bit product of $x$ and $y$.

Using the notations of [15], let us denote the class of depth-$d$ polynomial-size neural networks where the (integer) weights are polynomially bounded by $\widehat{LT}_d$ and the corresponding class where the weights are unrestricted by $LT_d$. It is easy to see that if iterated addition can be computed in $\widehat{LT}_2$, then multiplication can be computed in $\widehat{LT}_3$. We first prove the result on iterated addition. Our result hinges on a recent striking result of Goldmann, Håstad and Razborov [5]. The key observation is that iterated addition can be computed as a sum of polynomially many linear threshold ($LT_1$) functions (with exponential weights). Let us first state the result of Goldmann, Håstad and Razborov [5].

**Lemma 1**    [5] Let $\widetilde{LT}_d$ denote the class of depth-$d$ polynomial-size neural networks where the weights at the output gate are polynomially bounded integers (with no restriction on the weights of the other gates). Then $\widetilde{LT}_d = \widehat{LT}_d$ for any fixed integer $d \geq 1$.

The following lemma is a generalization of the result in [13]. Informally, the result says that if a function is 1 when a weighted sum (possibly exponential) of its inputs lies in one of polynomially many intervals, and is 0 otherwise, then the function can be computed as a sum of polynomially many $LT_1$ functions.

**Lemma 2**    Let $S = \sum_{i=1}^{n} w_i x_i$ and $f(X)$ be a function such that $f = 1$ if $S \in [l_i, u_i]$ for $i = 1, ..., N$ and $f = 0$ otherwise, where $N$ is polynomially bounded. Then $f$ can be computed as a sum of polynomially many $LT_1$ functions and thus $f \in \widetilde{LT}_2$.

Combining the above two lemmas yields a depth-2 neural network for iterated addition.

**Theorem 1**    Iterated addition is in $\widehat{LT}_2$.

It is also easy to see that iterated addition cannot be computed in $LT_1$. Simply observe that the first bit of the sum is the parity function, which does not belong to $LT_1$. Thus the above neural network for iterated addition has minimum possible depth.

**Theorem 2**    Multiplication of 2 $n$-bit integers can be computed in $\widehat{LT}_3$.

It follows from the results in [6] that the depth-3 neural network for multiplication stated in the above theorem has optimal depth.

We can further apply the results in [5] to construct small depth neural networks for division, powering and multiple product. Let us give a formal definition of these problems.

**Definition 3** Let X be an input $n$-bit integer $\geq 0$. We define *powering* to be the $n^2$-bit representation of $X^n$.

**Definition 4** Given $n$ $n$-bit integers $z_i$, $i = 1, ..., n$, We define *multiple product* to be the $n^2$-bit representation of $\prod_{i=1}^{n} z_i$.

Suppose we want to compute the quotient of two integers. Some quotient in binary representation might require infinitely many bits, however, a circuit can only compute the most significant bits of the quotient. If a number has both finite and infinite binary representation (for example $0.1 = 0.0111...$), we shall always express the number in its finite binary representation. We are interested in computing the truncated quotient, defined below:

**Definition 5** Let $X$ and $Y \geq 1$ be two input $n$ bit integers. Let $X/Y = \sum_{i=-\infty}^{n-1} z_i 2^i$ be the quotient of $X$ divided by $Y$. We define $\text{DIV}_k(X/Y)$ to be $X/Y$ *truncated* to the $(n+k)$-bit number, *i.e.*

$$\text{DIV}_k(X/Y) = \sum_{i=-k}^{n-1} z_i 2^i \qquad \square$$

In particular, $\text{DIV}_0(X/Y)$ is $\lfloor X/Y \rfloor$, the greatest integer $\leq X/Y$.

**Theorem 3**

1. Powering can be computed in $\widehat{LT}_3$.

2. $\text{DIV}_k(x/y)$ can be computed in $\widehat{LT}_3$.

3. Multiple Product can be computed in $\widehat{LT}_4$.

It can be shown from the lower-bound results in [9] that the neural networks for division are optimal in depth.

# References

[1] Y. S. Abu-Mostafa and D. Psaltis. Optical Neural Computers. *Scientific American* , 256(3):88–95, 1987.

[2] M. Ajtai. $\sum_1^1$-formulae on finite structures . *Annals of Pure and Applied Logic*, 24:1–48, 1983.

[3] A. K. Chandra, L. Stockmeyer, and U. Vishkin. Constant depth reducibility. *Siam J. Comput.*, 13:423–439, 1984.

[4] M. Furst, J. B. Saxe, and M. Sipser. Parity, Circuits and the Polynomial-Time Hierarchy. *IEEE Symp. Found. Comp. Sci.*, 22:260–270, 1981.

[5] M. Goldmann, J. Håstad, and A. Razborov. Majority Gates vs. General Weighted Threshold Gates. preprint, 1991.

[6] A. Hajnal, W. Maass, P. Pudlak, M. Szegedy, and G. Turan. Threshold circuits of bounded depth. *IEEE Symp. Found. Comp. Sci.*, 28:99–110, 1987.

[7] J. Håstad and M. Goldmann. On the power of small-depth threshold circuits. In *Proceedings of the 31st IEEE FOCS*, pp. 610-618, 1990.

[8] T. Hofmeister, W. Hohberg and S. Köhling . Some notes on threshold circuits and multiplication in depth 4. *Information Processing Letters*, 39:219–225, 1991.

[9] T. Hofmeister and P. Pudlák, A proof that division is not in $TC_2^0$. Forschungsbericht Nr. 447, 1992, Uni Dortmund.

[10] J. J. Hopfield. Neural Networks and physical systems with emergent collective computational abilities. *Proceedings of the National Academy of Sciences*, 79:2554–2558, 1982.

[11] J. L. McClelland D. E. Rumelhart and the PDP Research Group. *Parallel Distributed Processing: Explorations in the Microstructure of Cognition, vol. 1.* MIT Press, 1986.

[12] W. S. McCulloch and W. Pitts. A Logical Calculus of Ideas Immanent in Nervous Activity. *Bulletin of Mathematical Biophysics*, 5:115–133, 1943.

[13] S. Muroga. The principle of majority decision logic elements and the complexity of their circuits. *Intl. Conf. on Information Processing, Paris, France*, June 1959.

[14] K. Y. Siu and J. Bruck. Neural Computation of Arithmetic Functions. *Proc. IEEE*, 78, No. 10:1669–1675, October 1990. Special Issue on Neural Networks.

[15] K.-Y. Siu and J. Bruck. On the Power of Threshold Circuits with Small Weights . *SIAM J. Discrete Math.*, 4(3):423–435, August 1991.

[16] K.-Y. Siu, J. Bruck, T. Kailath, and T. Hofmeister. Depth-Efficient Neural Networks for Division and Related Problems . to appear in *IEEE Trans. Information Theory*, 1993.

[17] K.-Y. Siu and V. Roychowdhury. On Optimal Depth Threshold Circuits for Mulitplication and Related Problems. to appear in *SIAM J. Discrete Math.*

[18] A. Yao. Separating the polynomial-time hierarchy by oracles. *IEEE Symp. Found. Comp. Sci.*, pages 1–10, 1985.
